# CYCLES: A Simulation Tool for Studying Cyclic Neural Networks

## Michael T. Gately
### Texas Instruments Incorporated, Dallas, TX 75265

## ABSTRACT

A computer program has been designed and implemented to allow a researcher to analyze the oscillatory behavior of simulated neural networks with cyclic connectivity. The computer program, implemented on the Texas Instruments Explorer/Odyssey system, and the results of numerous experiments are discussed.

The program, CYCLES, allows a user to construct, operate, and inspect neural networks containing cyclic connection paths with the aid of a powerful graphics-based interface. Numerous cycles have been studied, including cycles with one or more activation points, non-interruptible cycles, cycles with variable path lengths, and interacting cycles. The final class, interacting cycles, is important due to its ability to implement time-dependent goal processing in neural networks.

## INTRODUCTION

Neural networks are capable of many types of computation. However, the majority of researchers are currently limiting their studies to various forms of mapping systems; such as content addressable memories, expert system engines, and artificial retinas. Typically, these systems have one layer of fully connected neurons or several layers of neurons with limited (forward direction only) connectivity. I have defined a new neural network topology; a two-dimensional lattice of neurons connected in such a way that circular paths are possible.

The neural networks defined can be viewed as a grid of neurons with one edge containing input neurons and the opposite edge containing output neurons [Figure 1]. Within the grid, any neuron can be connected to any other. Thus from one point of view, this is a multi-layered system with full connectivity. I view the weights of the connections as being the long term memory (LTM) of the system and the propagation of information through the grid as being it's short term memory (STM).

The topology of connectivity between neurons can take on any number of patterns. Using the mammalian brain as a guide, I have limited the amount of connectivity to something much less then total. In addition to making analysis of such systems less complex, limiting the connectivity to some small percentage of the total number of neurons reduces the amount of memory used in computer simulations. In general, the connectivity can be purely random, or can form any of a number of patterns that are repeated across the grid of neurons.

The program CYCLES allows the user to quickly describe the shape of the neural network grid, the source of input data, the destination of the output data, the pattern of connectivity. Once constructed, the network can be "run." during which time the STM may be viewed graphically.

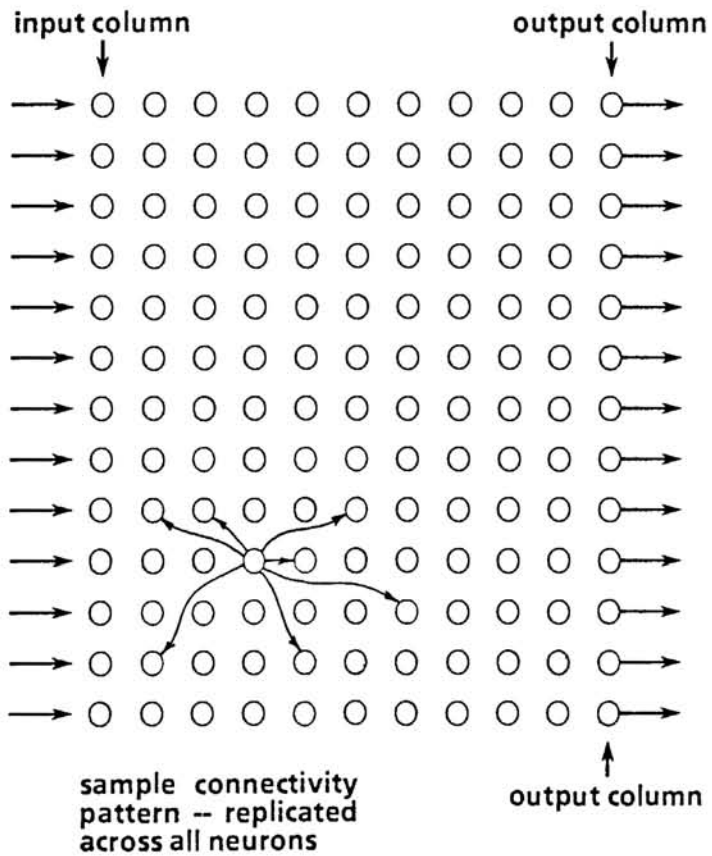

input column          output column

sample connectivity
pattern -- replicated
across all neurons

output column

Figure 1. COMPONENTS OF A CYCLES NEURAL NETWORK

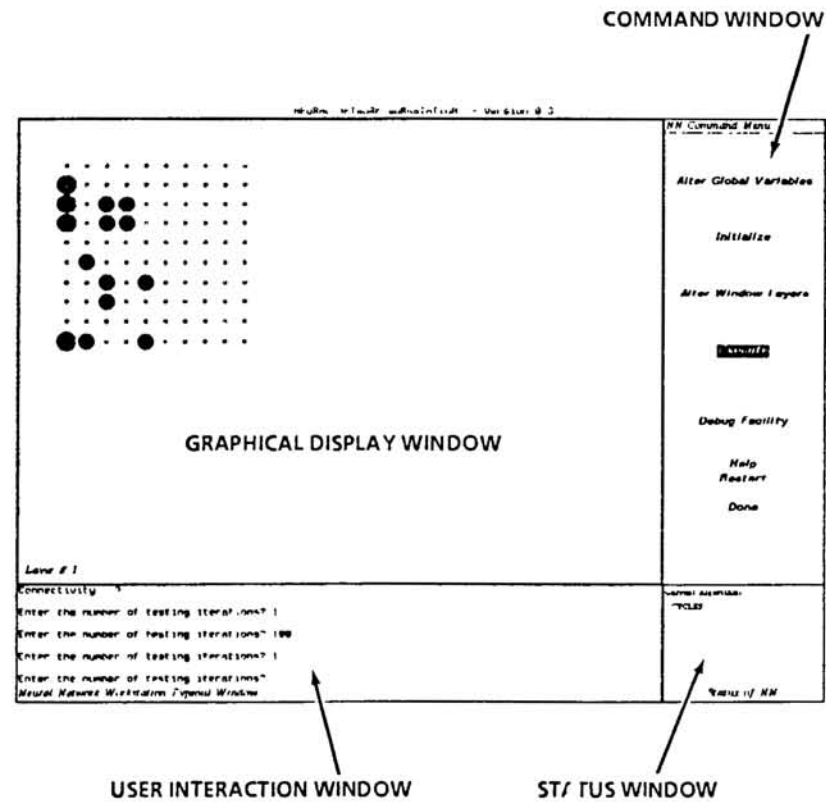

COMMAND WINDOW

GRAPHICAL DISPLAY WINDOW

USER INTERACTION WINDOW          STATUS WINDOW

Figure 2. NEURAL NETWORK WORKSTATION INTERFACE

## IMPLEMENTATION

CYCLES was implemented on a TI Explorer/Odyssey computer system with 8MB of RAM and 128MB of Virtual Memory. The program was written in Common LISP. The program was started in July of 1986, put aside for a while, and finished in March of 1987. Since that time, numerous small enhancements have been made – and the system has been used to test various theories of cyclic neural networks.

The code was integrated into the Neural Network Workstation (NNW), an interface to various neural network algorithms. The NNW utilizes the window interface of the Explorer LISP machine to present a consistent command input and graphical output to a variety of neural network algorithms [Figure 2].

The backpropagation-like neurons are collected together into a large three-dimensional array. The implementation actually allows the use of multiple two-dimensional grids; to date, however, I have studied only single-grid systems.

Each neuron in a CYCLES simulation consists of a list of information; the value of the neuron, the time that the neuron last fired, a temporary value used during the computation of the new value, and a list of the neurons connectivity. The connectivity list stores the location of a related neuron and the strength of the connection between the two neurons. Because the system is implemented in arrays and lists, large systems tend to be very slow. However, most of my analysis has taken place on very small systems (< 80 neurons) and for this size the speed is acceptable.

To help gauge the speed of CYCLES, a single grid system containing 100 neurons takes 0.8 seconds and 1235 cons cells (memory cells) to complete one update within the LISP machine. If the graphics interface is disabled, a test requiring 100 updates takes a total of 10.56 seconds.

## TYPES OF CYCLES

As mentioned above, several types of cycles have been observed. Each of these can be used for different applications. Figure 3 shows some of these cycles.

1. SIMPLE cycles are those that have one or more points of activation traveling across a set number of neurons in a particular order. The path length can be any size.
2. NON-INTERRUPTABLE cycles are those that have sufficiently strong connectivity strengths that random flows of activation which interact with the cycle will not upset or vary the original cycle.
3. VARIABLE PATH LENGTH cycles can, based upon external information, change their path length. There must be one or more neurons that are always a part of the path.
4. INTERACTING cycles typically have one neuron in common. Each cycle must have at least one other neuron involved at the junction point in order to keep the cycles separate. This type of cycle has been shown to implement a complex form of a clock where the product of the two (or more) path lengths are the fundamental frequency.

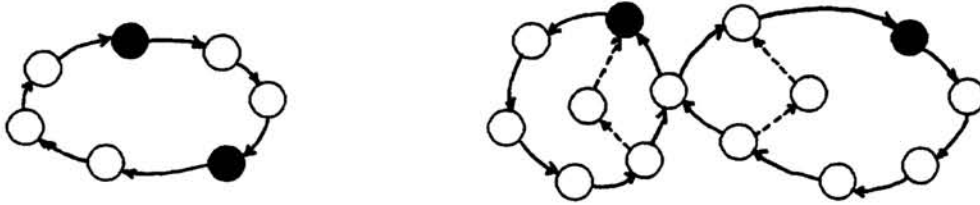

Figure 3. Types of Cycles [Simple and Interacting]

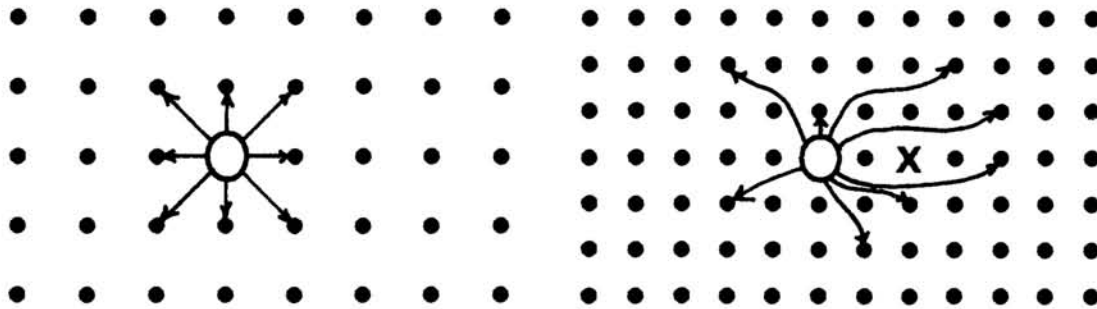

Figure 4. Types of Connectivity [Nearest Neighbor and Gaussian]

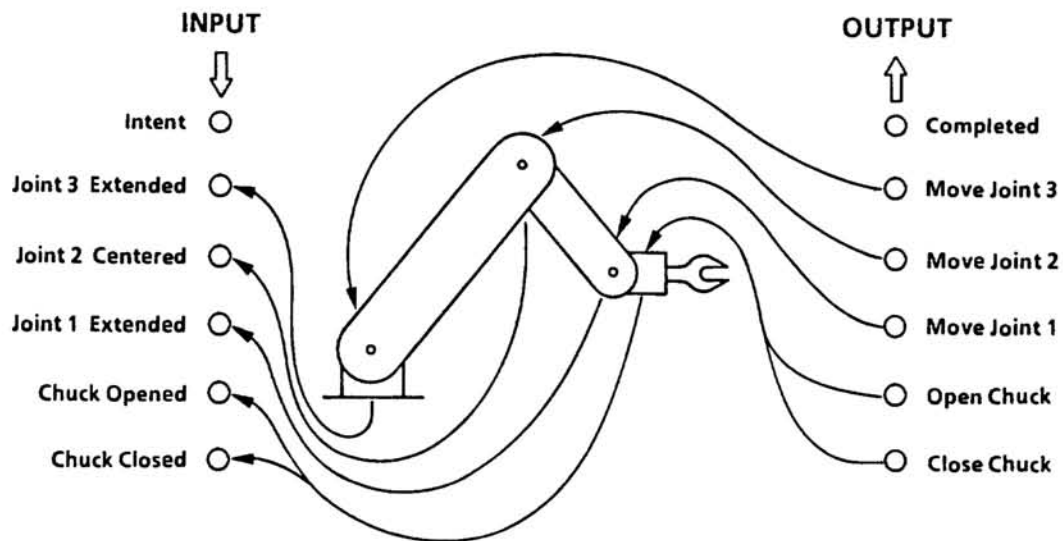

Figure 5. Robot Arm used in Example

## CONNECTIVITY

Several types of connectivity have been investigated. These are shown in Figure 4.

1. In TOTAL connectivity, every neuron is connected to every other neuron. This particular pattern produces very complex interactions with no apparent stability.

2. With RANDOM connectivity, each neuron is connected to a random number of other neurons. These other neurons can be anywhere in the grid.

3. A very useful type of connectivity is to have a PATTERN. The patterns can be of any shape, typically having one neuron feed its nearest neighbors.

4. Finally, the GAUSSIAN pattern has been used with the most success. In this pattern, each neuron is connected to a set number of nodes – but the selection is random. Further, the distribution of nodes is in a Gaussian shape, centered around a point "forward" of itself. Thus the flow of information, in general, moves forward, but the connectivity allows cycles to be formed.

## ALGORITHM

The algorithm currently being used in the system is a standard inner product equation with a sigmoidal threshold function. Each time a neuron's weight is to be calculated, the value of each contributing neuron on the connectivity list is multiplied by the strength of the connection and summed. This sum is passed through a sigmoidal thresholding function. The value of the neuron is changed to be the result of this threshold function. As you can see, the system updates neurons in an ordered fashion, thus certain interactions will not be observed. Since timing information is saved in the neurons, asynchrony could be simulated.

Initially, the weights of the connections are set randomly. A number of interesting cycles have been observed as a result of this randomness. However, several experiments have required specific weights. To accommodate this, an interface to the weight matrix is used. The user can create any set of connection strengths desired.

I have experimented with several learning algorithms–that is, algorithms that change the connection weights. The first mechanism was a simple Hebbian rule that states that if two neurons both fire, and there is a connection between them, then strengthen the strength of that connection. A second algorithm I experimented with used a pain/pleasure indicator to strengthen or weaken weights.

An algorithm that is currently under development actually presets the weights from a grammar of activity required of the network. Thus, the user can describe a process that must be controlled by a network using a simple grammar. This description is then "compiled" into a set of weights that contain cycles to indicate time-independent components of the activity.

## USAGE

Even without a biological background, it is easy to see that the processing power of the human brain is far more than present associative memories. Our repertoire of capabilities includes, among other things: memory of a time line, creativity, numerous types of biological clocks, and the ability to create and execute complex plans. The CYCLES algorithm has been shown to be capable of executing complex, time-variable plans.

A plan can be defined as a sequence of actions that must be performed in some preset order. Under this definition, the execution of a plan would be very straightforward. However, when individual actions within the plan take an indeterminate length of time, it is necessary to construct an execution engine capable of dealing with unexpected time delays. Such a system must also be able to abort the processing of a plan based on new data.

With careful programming of connection weights, I have been able to use CYCLES to execute time-variable plans. The particular example I have chosen is for a robot arm to change its tool. In this activity, once the controller receives the signal that the motion required, a series of actions take place that result in the tool being changed.

As input to this system I have used a number of sensors that may be found in a robot; extension sensors in 2-D joints and pressure sensors in articulators. The outputs of this network are pulses that I have defined to activate motors on the robot arm. Figure 5 shows how this system could be implemented. Figure 6 indicates the steps required to perform the task. Simple time delays, such as found with binding motors and misplaced objects are accommodated with the built in time-independence.

The small cycles that occur within the neural network can be thought of as short term memory. The cycle acts as a place holder – keeping track of the system's current place in a series of tasks. This type of pausing is necessary in many "real" activities such as simple process control or the analysis of time varying data.

## IMPLICATIONS

The success of CYCLES to simple process control activities such as robot arm control implies that there is a whole new area of applications for neural networks beyond present associative memories. The exploitation of the flow of activation as a form of short term memory provides us with a technique for dealing with many of the "other" type of computations which humans perform.

The future of the CYCLES algorithm will take two directions. First, the completion of a grammar and compiler for encoding process control tasks into a network. Second, other learning algorithms will be investigated which are capable of adding and removing connections and altering the strengths of connections based upon an abstract pain/pleasure indicator.

The robot gets a signal to begin the tool change process. A cycle is started that outputs a signal to the chuck motor.

When sensor indicates that the chuck is open, the first cycle is stopped and a second begins activating the motor in the first joint.

When the first joint is fully extended, the joint sensor sends a signal that stops that cycle, and begins one that outputs a signal to the second joint.

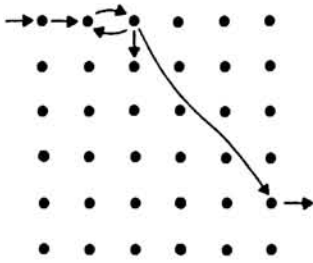
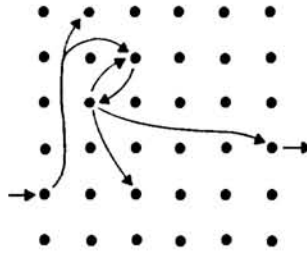
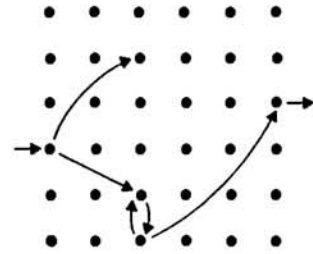

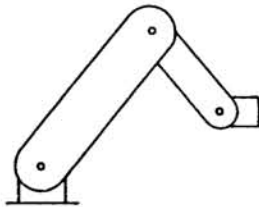
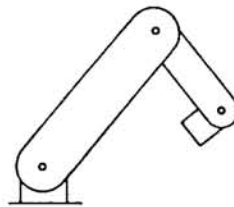
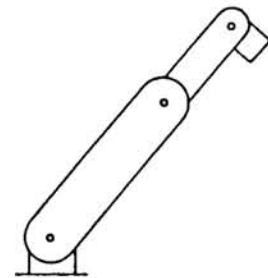

When the joint indicator indicates that the joint is centered, it changes the flow of activation to cause a cycle that activates the third joint.

Next, the chuck is closed around the new tool bit.

The last signal ends the sequence of cycles and sends the completed signal.

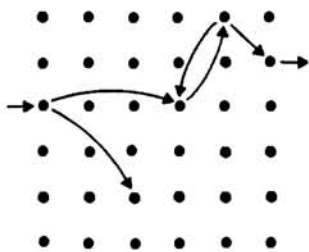
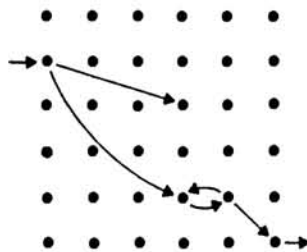
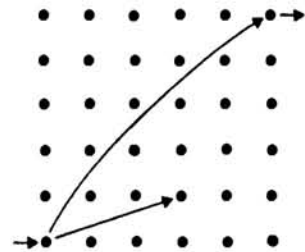

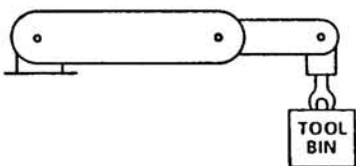
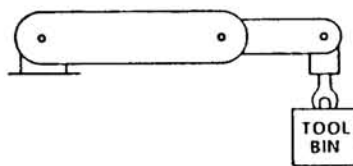
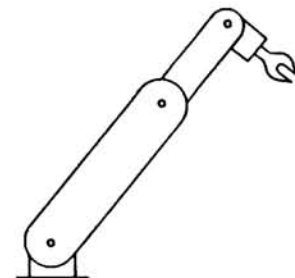

Figure 6. Example use of CYCLES to control a Robot Arm